# Neuron-MOS Temporal Winner Search Hardware for Fully-Parallel Data Processing

**Tadashi SHIBATA, Tsutomu NAKAI, Tatsuo MORIMOTO**
**Ryu KAIHARA, Takeo YAMASHITA, and Tadahiro OHMI**
Department of Electronic Engineering
Tohoku University
Aza-Aoba, Aramaki, Aobaku, Sendai 980-77 JAPAN

## Abstract

A unique architecture of winner search hardware has been developed using a novel neuron-like high functionality device called Neuron MOS transistor (or $\nu$MOS in short) [1,2] as a key circuit element. The circuits developed in this work can find the location of the maximum (or minimum) signal among a number of input data on the continuous-time basis, thus enabling real-time winner tracking as well as fully-parallel sorting of multiple input data. We have developed two circuit schemes. One is an ensemble of self-loop-selecting $\nu$MOS ring oscillators finding the winner as an oscillating node. The other is an ensemble of $\nu$MOS variable threshold inverters receiving a common ramp-voltage for competitive excitation where data sorting is conducted through consecutive winner search actions. Test circuits were fabricated by a double-polysilicon CMOS process and their operation has been experimentally verified.

## 1 INTRODUCTION

Search for the largest (or the smallest) among a number of input data, i.e., the winner-take-all (WTA) action, is an essential part of intelligent data processing such as data retrieval in associative memories [3], vector quantization circuits [4], Kohonen's self-organizing maps [5] etc. In addition to the maximum or minimum search, data sorting also plays an essential role in a number of signal processing such as median filtering in image processing, evolutionary algorithms in optimizing problems [6] and so forth. Usually such data processing is carried out by software running on general purpose computers, but the computation time increases explo-

sively with the increase in the volume of data. In order to build electronic systems having a real-time-response capability, the direct implementation of fully parallel algorithms on the integrated circuits hardware is critically demanded.

A variety of WTA [4, 7, 8] circuits have been implemented so far based on analog current-mode circuit technologies. A number of cells, each composed of a current source, competitively share the total current specified by a global current sink and the winner is identified through the current concentration toward the cell via tacit positive feedback mechanisms. The circuit implementations using MOSFET's operating in the subthreshold regime [4, 7] are ideal for large scale integration due to its ultra low power nature. Although they are inherently slow at circuit levels, the performance at a system level is far superior to digital counterparts owing to the flexible computing algorithms of analog. In order to achieve a high speed operation, MOSFET's biased at strong inversion is also utilized in Ref. [8]. However, cost must be traded off for increased power.

What we are presenting in this paper is a unique WTA architecture implemented by $\nu$MOS technology [1,2]. In $\nu$MOS circuits the summation of multiples of voltage signals is conducted on the $\nu$MOS floating gate (or better be called "temporary floating gate" when used in a clocked scheme [9]) via charge sharing among capacitors, and the result of the summation controls the transistor action. The voltage-mode summation capability of $\nu$MOS has been uniquely utilized to produce the WTA action. No DC current flows for the sum operation itself in contrast to the Kirchhoff sum. In $\nu$MOS transistors, however, DC current flows in a CMOS inverter configuration when the floating gate is biased in the transition region. Therefore the power consumption is larger than in the subthreshold circuitries. However, the $\nu$MOS WTA's presented in this article will give an opportunity of high speed operation at much less power consumption than current-mode circuitries operating in the strong inversion mode. In the following we present two kinds of winner search hardware featuring very fast operation. The winner can be tracked in a continuous-time regime with a detection delay time of about 100psec, while the sorting of multiple data is conducted in a fixed frame of time of about 100nsec.

## 2   NEURON-MOS CONTINUOUS-TIME WTA

Fig. 1(a) shows a schematic circuit diagram of a $\nu$MOS continuous-time WTA for four input signals. Each signal is fed to an input-stage $\nu$MOS inverter-A: a

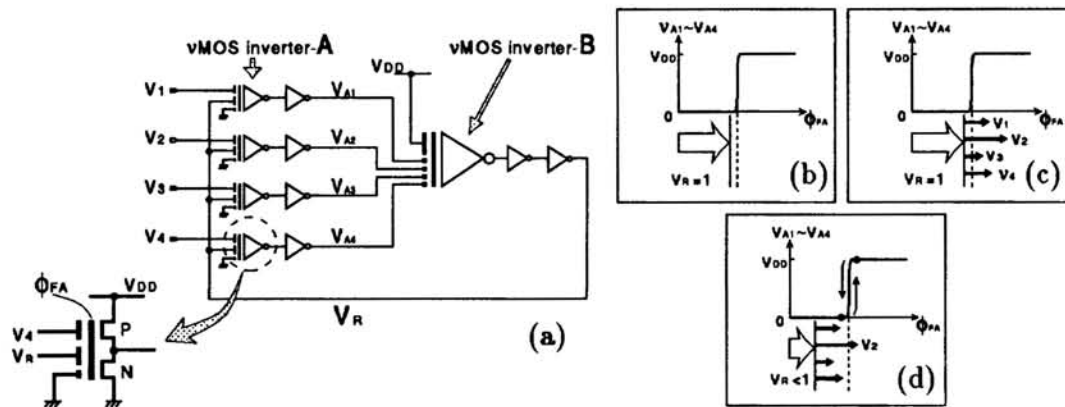

Figure 1: (a) Circuit diagram of $\nu$MOS continuous-time WTA circuit. (b)~(d) Response of $V_{A1} \sim V_{A4}$ as a function of the floating-gate potential of $\nu$MOS inverter-A.

CMOS inverter in which the common gate is made floating and its potential $\phi_{FA}$ is determined via capacitance coupling with three input terminals. $V_1 (\sim V_4)$ and $V_R$ are equally coupled to the floating gate and a small capacitance pulls down the floating gate to ground. The $\nu$MOS inverter-B is designed to turn on when the number of 1's in its inputs ($V_{A1} \sim V_{A4}$) is more than 1. When a feedback loop is formed as shown in the figure, it becomes a ring oscillator composed of odd-numbers of inverter stages.

When $V_1 \sim V_4 = 0$, the circuit is stable with $V_R = 1$ because inverter-A's do not turn on. This is because the small grounded capacitor pulls down the floating gate potential $\phi_{FA}$ a little smaller than its inverting threshold ($V_{DD}/2$) (see Fig. 1(b)). If non-zero signals are given to input terminals, more-than-one inverter-A's turn on (see Fig. 1(c)) and the inverter-B also turns on, thus initiating the transition of $V_R$ from $V_{DD}$ to 0. According to the decrease in $V_R$, some of the inverter-A's turn off but the inverter-B (number 1 detector) still stays at on-state until the last inverter-A turns off. When the last inverter-A, the one receiving the largest voltage input, turns off, the inverter-B also turns off and $V_R$ begins to increase. As a result, ring oscillation occurs only in the loop including the largest-input inverter-A(Fig. 1(d)). In this manner, the winner is identified as an oscillating node. The inverter-B can be altered to a number "2" detector or a number "3" detector etc. by just reducing the input voltage to the largest coupling capacitor. Then it is possible for top two or top three to be winners.

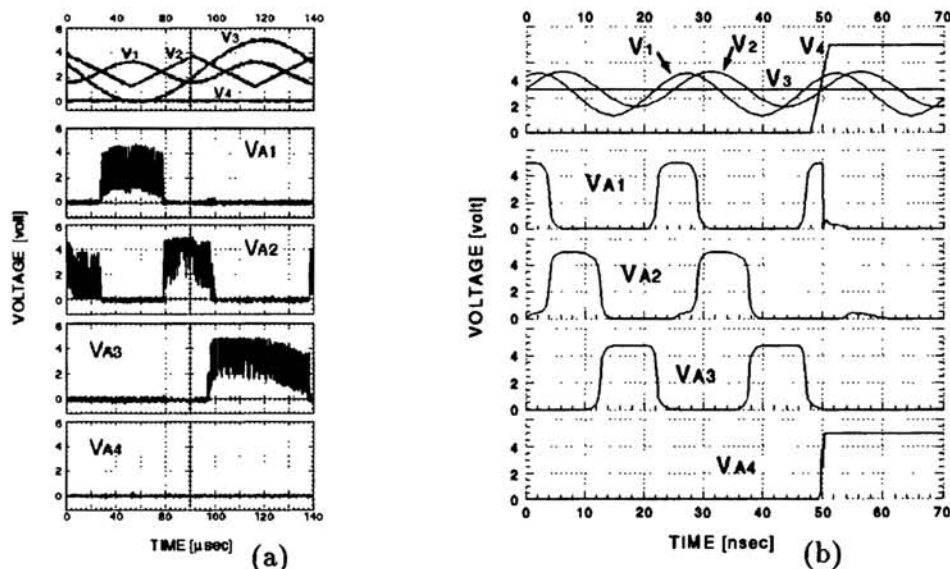

Figure 2: (a) Measured wave forms of four-input WTA as depicted in Fig. 1(a) (bread board experoment). (b) Simulation results for non-oscillating WTA explained in Fig. 3.

Fig. 2(a) demonstrates the measured wave forms of a bread-board test circuit composed of discrete components for verifying the circuit idea. It is clearly seen that ring oscillation occurs only at the temporal winner. However, the ring oscillation increases the power dissipation, and therefore, non-oscillating circuitry would be preferred. An example of simulation results for such a non-oscillating circuit is demonstrated in Fig. 2(b).

Fig. 3(a) gives the circuit diagram of a non-oscillating version of the $\nu$MOS

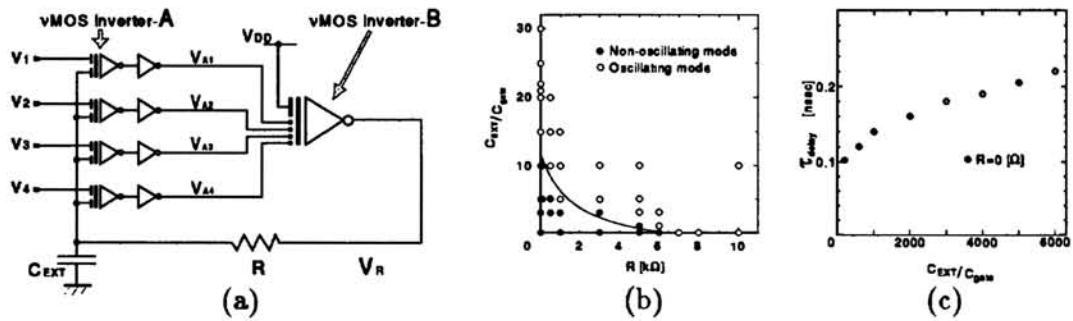

(a)       (b)       (c)

Figure 3: (a) Circuit diagram of non-oscillating-mode WTA. HSPICE simulation results: (b) combinations of $R$ and $C_{EXT}$ for non-oscillating mode; (c) winner detection delay as a function of capacitance load.

continuous-time WTA. In order to suppress the oscillation, the loop gain is reduced by removing the two-stage CMOS inverters in front of the inverter-B and $RC$ delay element is inserted in the feedback loop. The small grounded capacitors were removed in inverter-A's. The waveforms demonstrated in Fig. 2(b) are the HSPICE simulation results with $R = 0$ and $C_{EXT} = 20C_{gate}$ ($C_{gate}$: input capacitance of elemental CMOS inverter=5.16fF) . The circuit was simulated assuming a typical double-poly 0.5-$\mu$m CMOS process. Fig. 3(b) indicates the combinations of $R$ and $C_{EXT}$ yielding the non-oscillating mode of operation obtained by HSPICE simulation. It is important to note that if $C_{EXT} \geq 15C_{gate}$, non-oscillating mode appears with $R = 0$. This means the output resistance of the inverter-B plays the role of $R$. When the number of inverter-A's is increased, the increased capacitance load serves as $C_{EXT}$. Therefore, WTA having more than 19 input signals can operate in the non-oscillating mode. Fig. 3(c) represents the detection delay as a function of $C_{EXT}$. It is known that the increase in $C_{EXT}$, therefore the increase in the number of input signals to the WTA, does not significantly increase the detection delay and that the delay is only in the range of 100 to 200psec.

A photomicrograph of a test circuit of the non-oscillating mode WTA fabricated by Tohoku-University standard double-polysilicon CMOS process on 3-$\mu$m design rules, and the measurement results are shown in Fig. 4(a) and (b), respectively.

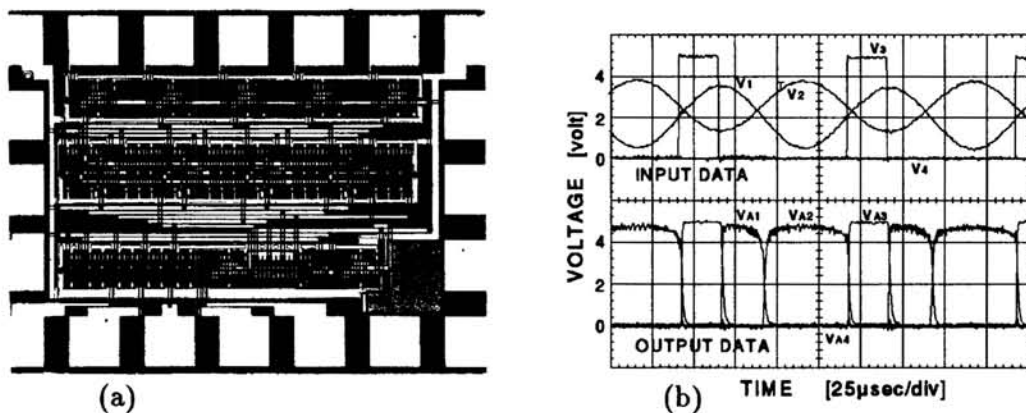

(a)       (b)    TIME    [25$\mu$sec/div]

Figure 4: (a) Photomicrograph of a test circuit for 4-input continuous-time WTA. Chip size is 800$\mu$m×500$\mu$m including all peripherals (3-$\mu$m rules). The core circuit of Fig. 3(a) occupies approximately 0.12 mm². (b) Measured wave forms.

## 3  NEURON-MOS DATA SORTING CIRCUITRY

The elemental idea of this circuit was first proposed at ISSCC '93 [3] as an application of the $\nu$MOS WTA circuit. In the present work, a clocked-$\nu$MOS technique [9] was introduced to enhance the accuracy and reliability of $\nu$MOS circuit operation and test circuits were fabricated and their operation have been verified.

Fig. 5(a) shows the circuit diagram of a test circuit for sorting three analog data $V_A$, $V_B$, and $V_C$, and a photomicrograph of a fabricated test circuit designed on 3-$\mu$m rules is shown in Fig. 5(b). Each input stage is a $\nu$MOS inverter: a CMOS inverter in which the common gate is made floating and its potential $\phi_F$ is determined by two input voltages via equally-weighted capacitance coupling, namely $\phi_F = (V_A + V_{RAMP})/2$. The reset signal forces the floating node be grounded, thus cancelling the charge on the $\nu$MOS floating gate each time before sorting. This is quite essential in achieving long-term reliability of $\nu$MOS operation. In the second stage are flip-flop memory cells to store sorting results. The third stage is a circuit which counts the number of 1's at its three input terminals and outputs the result in binary code. The concept of the $\nu$MOS A/D converter design [10] has been utilized in the circuit.

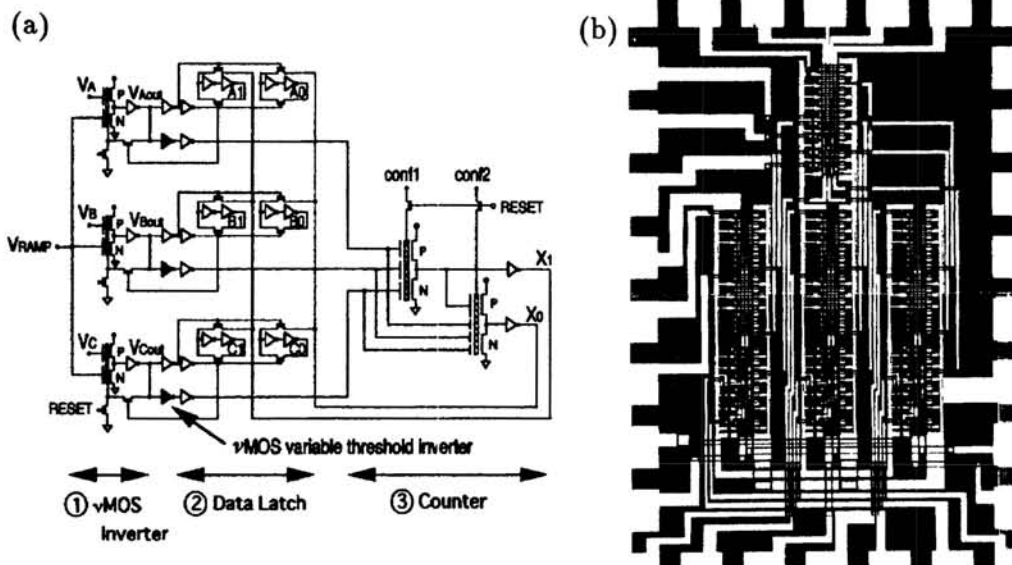

Figure 5: (a) Circuit diagram of $\nu$MOS data-soring circuit. (b)Photomicrograph of a test circuit fabricated by Tohoku Univ. Standard double-polysillicon CMOS process (3-$\mu$m rules). Chip size is $1250\mu$m$\times800\mu$m including all peripherals.

The sorting circuit is activated by ramping up $V_{RAMP}$ from 0V to $V_{DD}$. Then the $\nu$MOS inverter receiving the largest input turns on first and the output data of the counter at this moment (0,0) is latched in the respective memory cells. The counter output changes to (0,1) after gate delays in the counter and this code is latched when the $\nu$MOS inverter receiving the second largest turns on. Then the counter counts up to (1,0). In this manner, the all input data are numbered according to the order of their magnitudes after a ramp voltage scan is completed.

The measurement results are demonstrated in Fig. 6(a) in comparison with the HSPICE simulation results. Simulation was carried out on the same architecture circuit designed on 0.5-$\mu$m design rules and operated under 3V power supply. For three analog input voltages: $V_A = 5V$, $V_B = 4V$, and $V_C = 2V$, (0,0), (0,1),

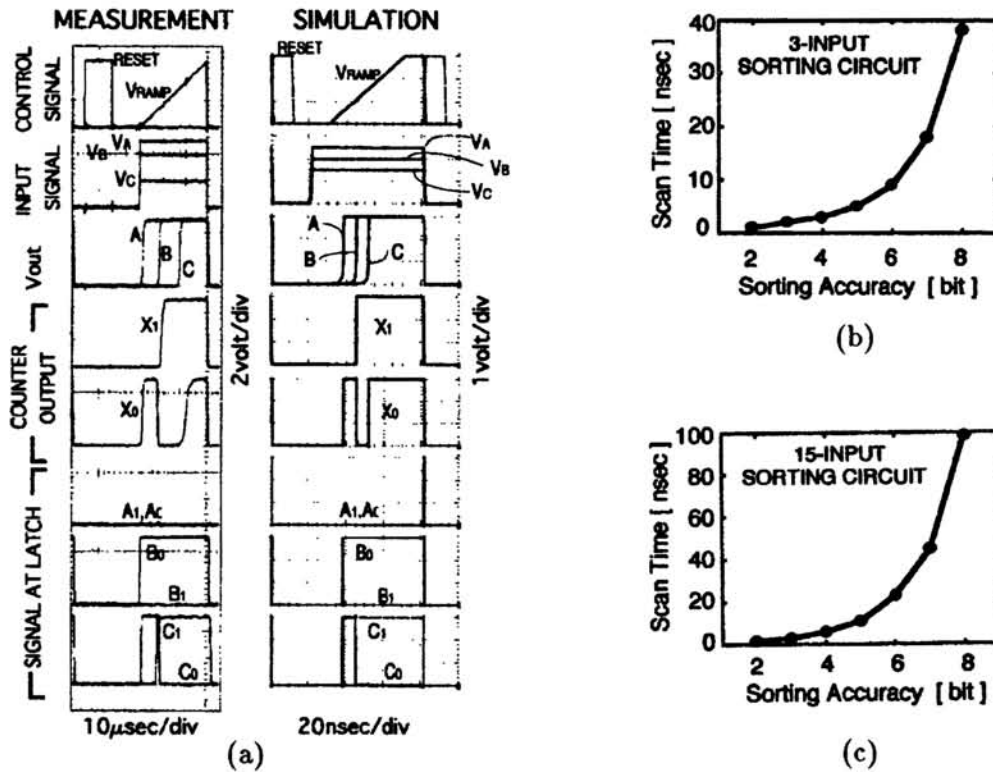

Figure 6: (a) Wave forms of the test circuit shown in Fig. 5(a) measured without buffer circuitry (left) and simulation results of a circuit designed with 0.5-$\mu$m rules (right). (b) Minimum scan time vs. sorting accuracy for a three-input sorter. (c) Minimum scan time vs. sorting accuracy for a 15-input sorter.

and (1,0) are latched, respectively, after the ramp voltage scan, thus accomplishing correct sorting. Slow operation of the test circuit is due to the loading effect caused by the direct probing of the node voltage without output buffer circuitries. The simulation with a 0.5-$\mu$m-design-rule circuit indicates the sorting is accomplished within the scan time of 40nsec.

In Fig. 6(b), the minimum scan time obtained by simulation is plotted as a function of the bit accuracy in sorting analog data. N-bit accuracy means the minimum voltage difference required for winner discrimination is $V_{DD}/2^2$. If the ramp rate is too fast, the $\nu$MOS inverter receiving the next largest data turns on before the correct counting results become available, leading to an erroneous operation. The scan time/accuracy relation in Fig. 6(b) is primarily determined by the response delay in the counter. It should be noted that the number of inverter stages in the counter ($\nu$MOS A/D converter) is always three indifferent to the number of output bits, namely, the delay would not increase significantly by the increase in the number of input data. In order to investigate this, a 15-input counter was designed and the delay time was evaluated by HSPICE simulation. It was 312 psec in comparison with 110 psec of the 3-input counter of Fig. 5(a). The scan time/accuracy relation for the 15-input sorting circuit is shown in Fig. 6(c), indicating the sorting of 15 input data can be accomplished in 100 nsec with 8-bit accuracy.

## 4  CONCLUSIONS

A novel neuron-like functional device $\nu$MOS has been successfully utilized in constructing intelligent electronic circuits which can carry out search for the temporal winner. As a result, it has become possible to perform data sorting as well as winner search in an instance, both requiring very time-consuming sequential data processing on a digital computer. The hardware algorithms presented here are typical examples of the $\nu$MOS binary-multivalue-analog merged computation scheme, which would play an important role in the future flexible data processing.

### Acknowledgements

This work was partially supported by Grant-in-Aid for Scientific Research (06402038) from the Ministry of Education, Science, Sports, and Culture, Japan. A part of this work was carried out in the Super Clean Room of Laboratory for Electronic Intelligent Systems, Research Institute of Electrical communication, Tohoku University.

### References

[1] T. Shibata and T. Ohmi, "A functional MOS transistor featuring gate-level weighted sum and threshold operations," IEEE Trans. Electron Devices, Vol. 39, No. 6, pp.1444-1455 (1992).

[2] T. Shibata, K. Kotani, T. Yamashita, H. Ishii, H. Kosaka, and T. Ohmi, "Implementing interlligence on silicon using neuron-like functional MOS transistors," in *Advances in Neural Information Processing Systems 6* (San Francisco, CA: Morgan Kaufmann 1994) pp. 919-926.

[3] T. Yamashita, T. Shibata, and T. Ohmi, "Neuron MOS winner-take-all circuit and its application to associative memory," in *ISSCC Dig. Tech. Papers*, Feb. 1993, FA 15.2, pp. 236-237.

[4] G. Gauwenberghs and V. Pedroni, "A charge-based CMOS parallel analog vector quantizer," in *Advances in Neural Information Processing Systems 7* (Cambridge, MA: The MIT Press 1995) pp. 779-786.

[5] T. Kohonen, *Self-Organization and Associative Memory*, 2nd ed. (New York: Springer-Verlag 1988).

[6] M. Kawamata, M. Abe, and T. Higuchi, "Evolutionary digital filters," in *Proc. Int. Workshop on Intelligent Signal Processing and Communication Systems*, seoul, Oct., 1994, pp. 263-268.

[7] J. Lazzaro, S. Ryckebusch, M. A. Mahowald, and C. A. Mead, "Winner-Take-All networks of O(N) complexity," in *Advances in Neural Information Processing Systems 1* (San Mateo, CA: Morgan Kaufmann 1989) pp. 703-711.

[8] J. Choi and B. J. Sheu, "A high-precision VLSI winner-take-all circuit for self-organizing neural networks," IEEE J. Solid State Circuits, Vol. 28, No. 5, pp.576-584 (1993).

[9] K. Kotani, T. Shibata, M. Imai, and T. Ohmi, "Clocked-Neuron-MOS logic circuits employing auto-threshold-adjustment," in *ISSCC Dig. Technical Papers*, Feb. 1995, FA 19.5, pp. 320-321.

[10] T. Shibata and T. Ohmi, "Neuron MOS binary-logic integrated circuits: Part II, Simplifying techniques of circuit configuration and their practical applications," IEEE Trans. Electron Devices, Vol. 40, No. 5, 974-979 (1993).
